# Sensory Modality Segregation

**Virginia R. de Sa**
Department of Cognitive Science
University of California, San Diego
La Jolla, CA 92093-0515
desa@ucsd.edu

## Abstract

Why are sensory modalities segregated the way they are? In this paper we show that sensory modalities are well designed for self-supervised cross-modal learning. Using the Minimizing-Disagreement algorithm on an unsupervised speech categorization task with visual (moving lips) and auditory (sound signal) inputs, we show that very informative auditory dimensions actually harm performance when moved to the visual side of the network. It is better to throw them away than to consider them part of the "visual input". We explain this finding in terms of the statistical structure in sensory inputs.

## 1   Introduction

In previous work [1, 2] we developed a simple neural network algorithm that learned categories from co-occurences of patterns to different sensory modalities. Using only the co-occuring patterns of lip motion and acoustic signal, the network learned *separate* visual and auditory networks (subnets) to distinguish 5 consonant vowel utterances. It performed almost as well as the corresponding supervised algorithm, where the utterance label is given, on the same data and significantly better than a strategy of separate unsupervised clustering in each modality followed by clustering of these clusters (This strategy is used to initialize our algorithm).

In this paper we show that the success of this biologically motivated algorithm depends crucially on the statistics of features derived from different sensory modalities. We do this by examining the performance when the two "network-modalities" or pseudo-modalities are made up of inputs from the different sensory modalities.

### The Minimizing-Disagreement Algorithm

The Minimizing-Disagreement (M-D) algorithm is designed to allow two (or more) modalities (or subnets) to simultaneously train each other by finding a local minimum of the number of times the individual modalities disagree on their classification decision (see Figure 1). The modalities are essentially trained by running Kohonen's LVQ2.1 algorithm[3] but with the target class set by the output of the subnet of the other modality (receiving a co-occuring pattern) *not* a supervisory external signal. The steps of the algorithm are as follows.

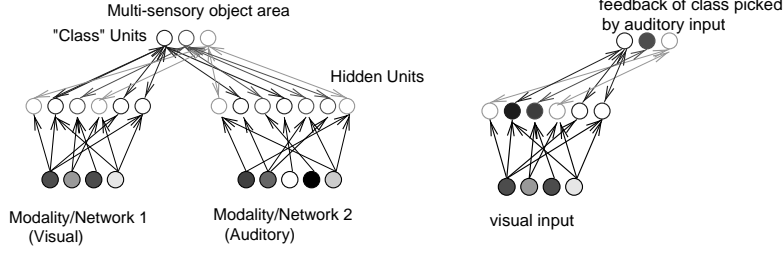

Figure 1: **The network for Minimizing-Disagreement algorithm.** The weights from the hidden units to the output units determine the "labels" of the hidden units. These weights are updated throughout training to allow hidden units to change classes if needed. During training each modality creates an output label for the other as shown on the right side of the figure. After training, each modality subnet is tested separately.

1. Initialize hidden unit weight vectors in each modality (unsupervised clustering)

2. Initialize hidden unit labels using unsupervised clustering of the activity patterns across the hidden units from both modalities

3. Repeat for each presentation of input patterns $X_1(n)$ and $X_2(n)$ to their respective modalities

   - For each modality, find the two nearest hidden unit weight vectors to the respective input pattern

   - Find the hypothesized output class in each modality (as given by the label of the hidden unit with closest weight vector). The label of a hidden unit is the output unit to which it projects most strongly.

   - For each modality update the hidden unit weight vectors according to the LVQ2.1 rule (Only the rules for modality 1 are given below)

     Updates are performed only if the current pattern $X_1(n)$ falls within c(n) of the border between two hidden units of different classes (one of them agreeing with the output from the other modality). In this case

     $$\vec{w}_{1_{i^*}}(n) = \vec{w}_{1_{i^*}}(n-1) + \varepsilon(n)\frac{(X_1(n) - \vec{w}_{1_{i^*}}(n-1))}{||X_1(n) - \vec{w}_{1_{i^*}}(n-1)||}$$

     $$\vec{w}_{1_{j^*}}(n) = \vec{w}_{1_{j^*}}(n-1) - \varepsilon(n)\frac{(X_1(n) - \vec{w}_{1_{j^*}}(n-1))}{||X_1(n) - \vec{w}_{1_{j^*}}(n-1)||}$$

     where $\vec{w}_{1_{i^*}}$ is the weight vector of the hidden unit with the same label, and $\vec{w}_{1_{j^*}}$ is the weight vector of the hidden unit with another label.

   - Update the labeling weights using Hebbian learning between the winning hidden unit and the output of the other modality.

In order to discourage runaway to one of the trivial global minima of disagreement, where both modalities only ever output one class, weights to the output class neurons are renormalized at each step. This normalization means that the algorithm is not modifying the output weights to minimize the disagreement but instead clustering the hidden unit representation using the output class given by the other modality. This objective is better for these weights as it balances the goal of agreement with the desire to avoid the trivial solution of all hidden units having the same label.

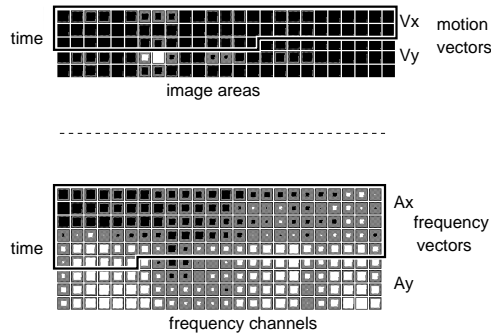

Figure 2: **An example Auditory and Visual pattern vector.** The figure shows which dimensions went into each of Ax, Ay, Vx, and Vy.

## 2 Experiments

### 2.1 Creation of Sub-Modalities

The original auditory and visual data were collected using an 8mm camcorder and directional microphone. The speaker spoke 118 repetitions of /ba/, /va/, /da/, /ga/, and /wa/. The first 98 samples of each utterance class formed the training set and the remaining 20 the test set. The auditory feature vector was encoded using a 24 channel mel code[1] over 20 msec windows overlapped by 10 msec. This is a coarse short time frequency encoding, which crudely approximates peripheral auditory processing. Each feature vector was linearly scaled so that all dimensions lie in the range [-1,1]. The final auditory code is a (24 × 9) 216 dimension vector for each utterance. An example auditory feature vector is shown in Figure 2 (bottom).

The visual data were processed using software designed and written by Ramprasad Polana [4]. Visual frames were digitized as 64 × 64 8 bit gray-level images using the Datacube MaxVideo system. Segments were taken as 6 frames before the acoustically determined utterance offset and 4 after. The normal flow was computed using differential techniques between successive frames. Each pair of frames was then averaged and then these averaged frames were divided into 25 equal areas (5 × 5) and the motion magnitudes within each frame were averaged within each area. The final visual feature vector of dimension (5 frames × 25 areas) 125 was linearly normalized as for the auditory vectors. An example visual feature vector is shown in Figure 2 (top).

The original auditory and visual feature vectors were divided into two parts (called Ax, Ay and Vx,Vy as shown in Figure 2). The partition was arbitrarily determined as a compromise between wanting a similar number of dimensions and similar information content in each part. (We did not search over partitions; the experiments below were performed only for this partition). Our goal is to combine them in different ways and observe the performance of the minimizing-disagreement algorithm.

We first benchmarked the divided "sub-modalities" to see how useful they were for the task. For this, we ran a supervised algorithm on each subset. The performance measurements are shown in Table 1.

| Sub-Modality | Supervised Performance |
|---|---|
| Ax | $89 \pm 2$ |
| Ay | $91 \pm 2$ |
| Vx | $83 \pm 2$ |
| Vy | $77 \pm 3$ |

Table 1: **Supervised performance of each of the sub-modalities**. All numbers give percent correct classifications on independent test sets $\pm$ standard deviations.

## 2.2   Creation of Pseudo-Modalities

Pseudo-modalities were created by combining all combinations (of 3 or less) of Ax, Ay, Vx and Vy; thus Ax+Vx+Vy (Ax+V) would be a pseudo-modality. The idea is to test all possible combinations of pseudo-modalities and compare the resulting performance of the final individual subnets with what a supervised algorithm could do with the same dimensions.

## 2.3   Pseudo-Modality Experiments

In order to allow fair comparison, appropriate parameters were found for each modality division. The data were divided into 75% Training, and 25% Test data. Optimal parameters were selected by observing performance on the training data, and performance is reported on the test data.

The results for all possible divisions are presented in Figure 3. Each network has the following key. The light gray bar and number represents the test-set performance of the pseudo-modality consisting of the sub-modalities listed below it. The darker bar and number represents the test-set performance of the other pseudo-modality. The black outlines (and numbers above the outline) give the performance of the corresponding supervised algorithm (LVQ2.1) with the same data. Thus, the empty area between the shaded area and black outline represents the loss from lack of supervision.

Looking at the figure, one can make several comparisons. For each submodality, we can ask: To get the best performance of a subnet using those dimensions, where should one put the other sub-modalities in a M-D network? For instance, to answer that question of Ax, one would compare the performance of the Ax subnet in Ax/Ay+V network with that of the Ax+Ay subnet in the Ax+Ay/Vx+Vy network, with that of the Ax+Vx+Vy subnet in the Ax+Vx+Vy/Ay network etc. The subnet containing Ax that performs the best is the Ax+Ay subnet (trained with co-modality Vx+Vy). In fact, it turns out that for each submodality, the architecture for optimal post-training performance of the subnet containing that submodality, is to put the dimensions from the same "real" modality on the same side and those from the other modality on the other side.

This raises the question: Is performance better for the Ax+Ay/Vx+Vy network than the Ax/Ay+Vx+Vy network because the *benefit* of having Ay with Ax is greater than that of having Ay with Vx and Vy (in other words, are there some higher order relationships between dimensions in Ax and those in Ay that require both dimensions to be learned by the same subnet) OR is it actually harmful to have Ay on the opposite side from Ax? We can answer this question by comparing the performance of the Ax/Ay+Vx+Vy network with that of the Ax/Vx+Vy network as shown in Figure 4. For that particular division, the results are not significantly different (even though we have removed the most useful dimensions), but for all the other divisions, performance is improved when dimensions are *removed* so that only dimensions from one "real" sensory modality are on one side. For example, the two graphs in the second column show that it is actually *harmful* to include the very useful Ax dimensions on the visual side of the network – we do better when we

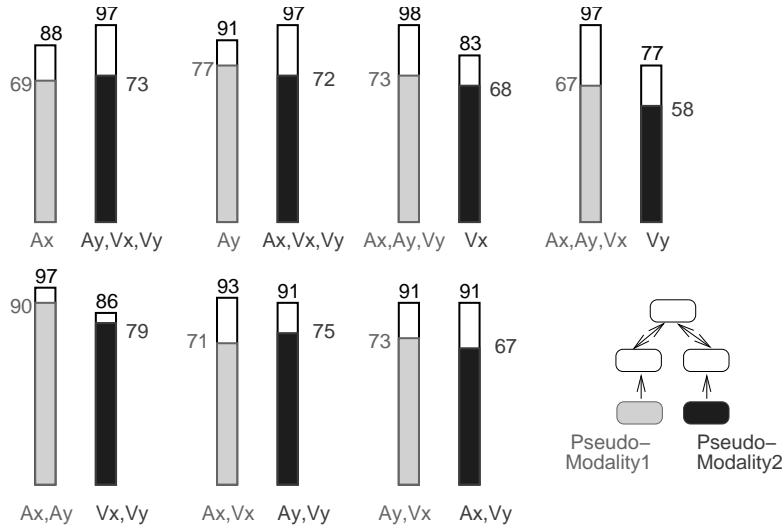

Figure 3: **Self-supervised and Supervised performances for the various pseudo-modality divisions**. Standard errors for the self-supervised performance means are $\pm 1$. Those for the supervised performances are $\pm .5$.

throw them away. Note that this is true even though a supervised network with Ax+Vx+Vy does much better than a supervised network with Vx+Vy — this is not a simple feature selection result.

## 2.4 Correlational structure is important

Why do we get these results? The answer is that the results are very dependent on the statistical structure between dimensions within and between different sensory modalities.

Consider a simpler system of two 1-Dimensional modalities and two classes of objects. Assume that the sensation detected by each modality has a probability density given by a Gaussian of different mean for each class. The densities seen by each modality are shown in Figure 5. In part A) of the Figure, the joint density for the stimuli to both modalities is shown for the case of conditionally uncorrelated stimuli (within each class, the inputs are uncorrelated). Parts C) and D) show the changing joint density as the sensations to the two modalities become more correlated within each class. Notice that the density changes from a "two blob" structure to more of a "ridge" structure. As it does this the projection of the joint density gives less indication of the underlying bi-modal structure *and* the local minimum of the Minimizing-Disagreement Energy function gets shallower and narrower. This means that the M-D algorithm would be less likely to find the correct boundary.

A more intuitive explanation is shown in Figure 6. In the figure imagine that there are two classes of objects, with densities given by the thick curve and the thin curve and that this marginal density is the same in each one-dimensional modality. The line drawing below the densities, shows two possible scenarios for how the "modalities" may be related. In the top case, the modalities are conditionally independent. Given that a "thick" object is present, the particular pattern to each modality is independent. The lines represent a possible sampling of data (where points are joined if they co-occured). The minimizing disagreement algorithm wants to find a line from top to bottom that crosses the fewest lines – within the pattern space, disagreement is minimized for the dashed line shown.

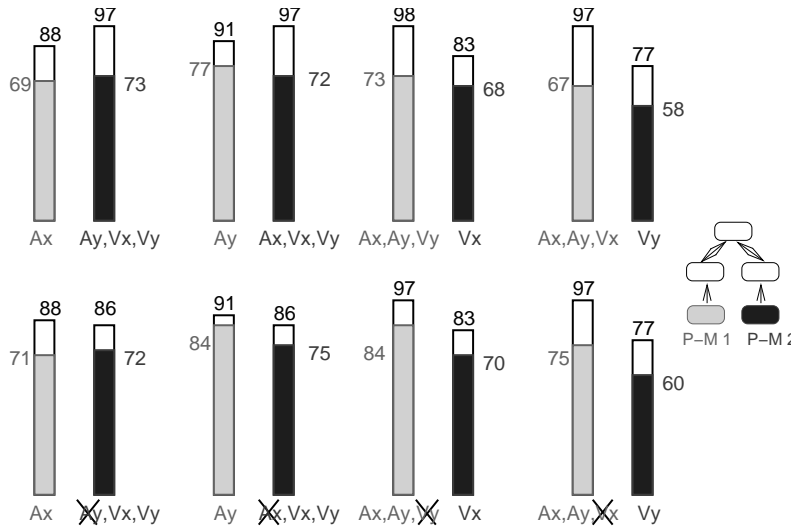

Figure 4: **This figure shows the benefits of having a pseudo-modality composed of dimensions from only ONE real modality (even if this means throwing away useful dimensions).** Standard errors for the self-supervised performance means are ±1. Those for the supervised performances are ±.5.

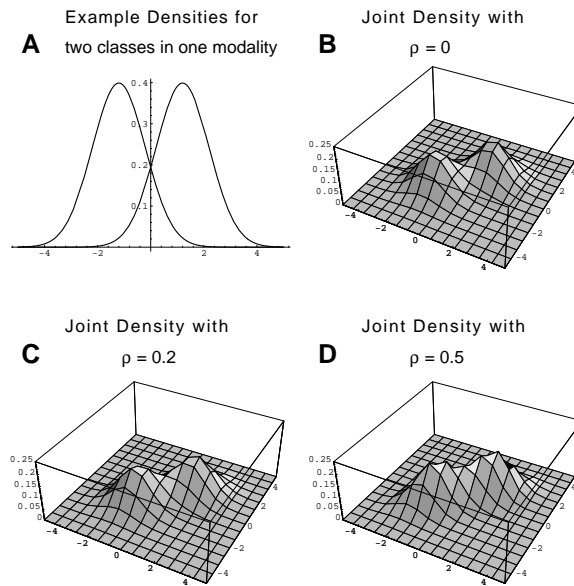

Figure 5: Different joint densities with the same marginal densities

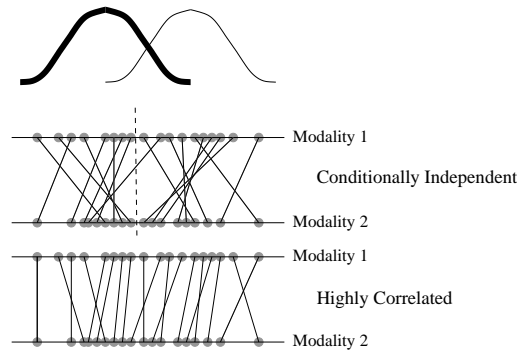

Figure 6: Lines are joined between co-occuring patterns in two imaginary 1-D modalities (as shown at top). The M-D algorithm wants to find a partition that crosses the fewest lines.

Conditional Information ($I(X;Y|Class)$)
(with diagonal zeroed)

Within-Class Correlation Coefficients
(averaged over each class)

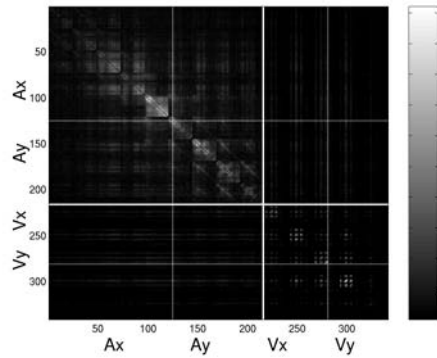
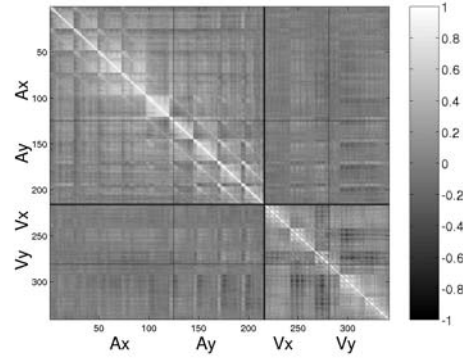

Figure 7: **Statistical Structure of our data**

In the bottom case, the modalities are strongly dependent. In this case there are many local minima for minimum disagreement, that are not closely related to the class boundary. It is easy for the networks to minimize the disagreement between the outputs of the modalities, without paying attention to the class. Having two very strongly dependent variables, one on each side of the network, means that the network can minimize disagreement by simply listening to those units.

To verify that our auditory-visual results were due to statistical differences between the dimensions, we examined the statistical structure of our data. It turns out that, within a class, the correlation coefficient between most pairs of dimensions is fairly low. However, for related auditory features (similar time and frequency band) correlations are high and also for related visual features. This is shown in Figure 7. We also computed the conditional mutual information between each pair of features given the class $I(x;y|Class)$. This is also shown in Figure 7. This value is 0 if and only if the two features are conditionally independent given the class. The graphs show that many of the auditory dimensions are highly dependent on each other (even given the class), as are many of the visual dimensions. This makes them unsuitable for serving on the other side of a M-D network.

## 2.5  Discussion

The minimizing-disagreement algorithm was initially developed as a model of self-supervised cortical learning and the importance of conditionally uncorrelated structure was mentioned in [5]. Since then people have used similar partly-supervised algorithms to deal with limited labeled data in machine learning problems [6, 7]. They have also emphasized the importance of conditional independence between the two sides of the input. However in the co-training style algorithms, inputs that are conditionally dependent are not helpful, but they are also not as harmful. Because the self-supervised algorithm is dependent on the class structure being evident in the joint space as its only source of supervision, it is very sensitive to conditionally dependent relationships between the modalities.

We have shown that different sensory modalities are ideally suited for teaching each other. Sensory modalities are also composed of submodalities (e.g. color and motion for the visual modality) which are also likely to be conditionally independent (and indeed may be actively kept so [8, 9, 10]). We suggest that brain connectivity may be constrained not only due to volume limits, but because limiting connectivity may be beneficial for learning.

### Acknowledgements

A preliminary version of this work appeared in a book chapter [5] in the book, Psychology of Learning and Motivation. This work is supported by NSF CAREER grant 0133996.

## Footnotes

[1]linear spacing below 1000 Hz and logarithmic above 1000 Hz.

# References

[1] Virginia R. de Sa. Learning classification with unlabeled data. In J.D. Cowan, G. Tesauro, and J. Alspector, editors, *Advances in Neural Information Processing Systems 6*, pages 112—119. Morgan Kaufmann, 1994.

[2] Virginia R. de Sa and Dana H. Ballard. Category learning through multimodality sensing. *Neural Computation*, 10(5):1097–1117, 1998.

[3] Teuvo Kohonen. Improved versions of learning vector quantization. In *IJCNN International Joint Conference on Neural Networks*, volume 1, pages I–545–I–550, 1990.

[4] Ramprasad Polana. *Temporal Texture and Activity Recognition*. PhD thesis, Department of Computer Science, University of Rochester, 1994.

[5] Virginia R. de Sa and Dana H. Ballard. Perceptual learning from cross-modal feedback. In R. L. Goldstone, P.G. Schyns, and D. L. Medin, editors, *Psychology of Learning and Motivation*, volume 36, pages 309–351. Academic Press, San Diego, CA, 1997.

[6] A. Blum and T. Mitchell. Combining labeled and unlabeled data with co-training. In *Proceedings of the Eleventh Annual Conference on Computational Learning Theory (COLT-98)*, pages 92–100, 1998.

[7] Ion Muslea, Steve Minton, and Craig Knoblock. Active + semi-supervised learning = robust multi-view learning. In *Proceedings of the 19th International Conference on Machine Learning (ICML 2002)*, pages 435–442, 2002.

[8] C. McCollough. Color adaptation of edge-detectors in the human visual system. *Science*, 149:1115–1116, 1965.

[9] P.C. Dodwell and G.K. Humphrey. A functional theory of the mccollough effect. *Psychological Review*, 1990.

[10] F. H. Durgin and D.R. Proffitt. Combining recalibration and learning accounts of contingent aftereffects. In *Proceedings of the annual meeting of the Psychonomic Society*.
